# Attractive People: Assembling Loose-Limbed Models using Non-parametric Belief Propagation

**Leonid Sigal**
Department of Computer Science
Brown University
Providence, RI 02912
ls@cs.brown.edu

**Michael Isard**
Microsoft Research Silicon Valley
Mountain View, CA 94043
misard@microsoft.com

**Benjamin H. Sigelman**
Department of Computer Science
Brown University
Providence, RI 02912
bhsigelm@cs.brown.edu

**Michael J. Black**
Department of Computer Science
Brown University
Providence, RI 02912
black@cs.brown.edu

## Abstract

*The detection and pose estimation of people in images and video is made challenging by the variability of human appearance, the complexity of natural scenes, and the high dimensionality of articulated body models. To cope with these problems we represent the 3D human body as a graphical model in which the relationships between the body parts are represented by conditional probability distributions. We formulate the pose estimation problem as one of probabilistic inference over a graphical model where the random variables correspond to the individual limb parameters (position and orientation). Because the limbs are described by 6-dimensional vectors encoding pose in 3-space, discretization is impractical and the random variables in our model must be continuous-valued. To approximate belief propagation in such a graph we exploit a recently introduced generalization of the particle filter. This framework facilitates the automatic initialization of the body-model from low level cues and is robust to occlusion of body parts and scene clutter.*

## 1 Introduction

Recent approaches to person detection and tracking exploit articulated body models in which the body is viewed as a kinematic tree in 2D [14], 2.5D [16, 23], or 3D [2, 5, 6, 19, 21] leading to a parametric state-space representation of roughly 25–35 dimensions. The high dimensionality of the resulting state-space has motivated the development of specialized stochastic search algorithms that either exploit the highly redundant dynamics of typical human motions [19], or use hierarchical sampling schemes to exploit the tree-structured nature of the model [5, 15]. These schemes have been effective for tracking people wearing increasingly complex clothing in increasingly complex cluttered backgrounds [21]. There are however a number of important shortcomings of these ap-

proaches. Hierarchical body models lead to "top-down" search algorithms that make it difficult to incorporate "bottom-up" information about salient body parts available from special-purpose detectors (e.g. face or limb detectors). As a result, few, if any, of the above methods deal with the problem of automatic initialization of the body model. Furthermore, the difficulty of incorporating bottom-up information means that the algorithms are brittle; that is, when they lose track of the body, they have no way to recover. Finally, the fully coupled kinematic model results in a computationally challenging search problem because the search space cannot be naturally decomposed.

To address these problems, we propose a "loose-limbed" body model in which the limbs are not rigidly connected but are rather "attracted" to each other (hence the title "Attractive People"). Instead of representing the body as a single 33-dimensional kinematic tree, each limb is treated quasi-independently with soft constraints between the position and orientation of adjacent parts. The model resembles a Push Puppet toy which has elastic connections between the limbs (Figure 1*a*).

This type of model is not new for finding or tracking articulated objects and dates back at least to Fischler and Elschlager's pictorial structures [9]. Variations on this type of model have been recently applied by Burl *et al.* [1], Felzenszwalb and Huttenlocher [8], Coughlan and Ferreira [3] and Ioffe and Forsyth [11, 17]. The main benefits are that it supports inference algorithms where the computational cost is linear rather than exponential in the number of body parts, it allows elegant treatment of occlusion, and it permits automatic initialization based on individually unreliable low-level body-part detectors [25].

The work described here, like the previous work above, exploits this notion of flexible "spring"-like constraints [8] defined over individually modeled body parts [11, 17, 23], though we extend the approach to locate the parts in 3-space rather than the 2-dimensional image plane. The body is treated as a graphical model [13], where each node in the graph corresponds to an independently parameterized body part. The spatial constraints between body parts are defined as directed edges in the graph. Each edge has an associated conditional distribution that models the probabilistic relationship between the parts. Each node in the graph also has a corresponding image likelihood function that models the probability of observing various image measurements conditioned on the position and orientation of the part. Person detection (or tracking) then exploits *belief propagation* [24] to estimate the belief distribution over the parameters which takes into account the constraints and the observations.

This graphical inference problem is carried out using a recently proposed method that allows the parameters of the individual parts to be modeled using *continuous-valued* random variables rather than the discrete variables used in previous approaches. This is vital in our problem setting, since the discretization used in [8] is impractical once the body is modeled in 3-space. Similar versions of the algorithm were independently introduced by Sudderth *et al.* [22] under the name of Non-parametric Belief Propagation (NBP) and by Isard [12] as the PAMPAS algorithm. We adopt the framework of Isard while making use of the Gibbs sampler introduced by Sudderth *et al.* The algorithm extends the flexibility of particle filters to the problem of belief propagation and, in our context, allows the model to cope with general constraints between limbs, permits realistic appearance models, and provides resilience to clutter.

We develop the loose-limbed model in detail, formulate the constraints between limbs using mixture models, and outline the inference method. Using images from calibrated cameras we illustrate the inference of 3D human pose with belief propagation. We simulate noisy, bottom-up, feature detectors for the limbs and show how the inference method can resolve ambiguities and cope with clutter. While our focus here is on static detection and pose estimation, the body model can be extended in time to include temporal constraints on the limb motion; we save tracking for future work.

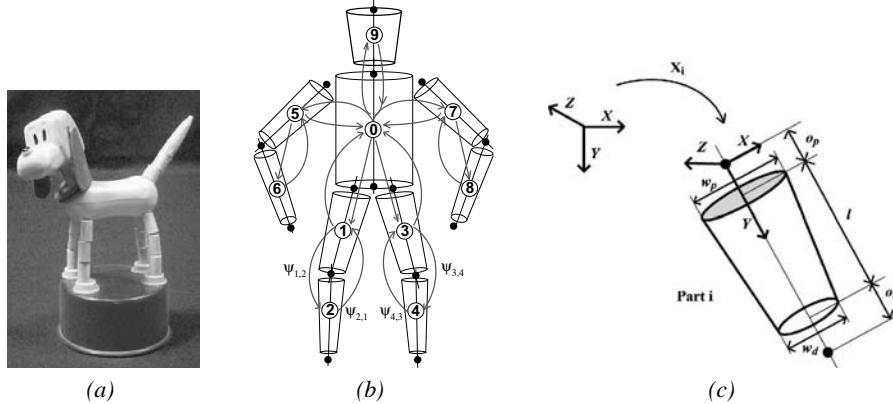

*(a)*          *(b)*          *(c)*

Figure 1: *(a)* Toy Push Puppet with elastic joints. *(b)* Graphical model for a person. Nodes represent limbs and arrows represent conditional dependencies between limbs. *(c)* Parameterization of part $i$.

## 2    A self-assembling body model

The body is represented by a graphical model in which each graph node corresponds to a body part (upper leg, torso, etc.). Each part has an associated configuration vector defining the part's position and orientation in 3-space. Placing each part in a global coordinate frame enables the part detectors to operate independently while the full body is assembled by inference over the graphical model. Edges in the graphical model correspond to spatial and angular relationships between adjacent body parts, as illustrated in Figure 1*b*. As is standard for graphical models we assume the variables in a node are conditionally independent of those in non-neighboring nodes given the values of the node's neighbors[1].

Each part/limb is modeled by a tapered cylinder having 5 fixed (person specific) and 6 estimated parameters. The fixed parameters $\Phi_i = (l_i, w_i^p, w_i^d, o_i^p, o_i^d)$ correspond respectively to the part length, width at the proximal and distal ends and the offset of the proximal and distal joints along the axis of the limb as shown in Figure 1*c*. The estimated parameters $\mathbf{X}_i^T = (\mathbf{x}_i^T, \Theta_i^T)$ represent the configuration of the part $i$ in a global coordinate frame where $\mathbf{x}_i \in \mathbb{R}^3$ and $\Theta_i \in \mathrm{SO}(3)$ are the 3D position of the proximal joint and the angular orientation of the part respectively. The rotations are represented by unit quaternions.

Each directed edge between parts $i$ and $j$ has an associated conditional distribution $\psi_{ij}(\mathbf{X}_i, \mathbf{X}_j)$ that encodes the compatibility between pairs of part configurations; that is, it models the probability of configuration $\mathbf{X}_j$ of part $j$ conditioned on the $\mathbf{X}_i$ of part $i$. For notational convenience we define an ordering on body parts going from the torso out towards the extremities and refer to conditionals that go along this ordering as "forward" conditionals. Conversely, the conditionals that go from the extremities towards the torso are referred to as "backward" conditionals. These intuitively correspond to kinematic and inverse-kinematic constraints respectively.

Conditional distributions were constructed by hand to capture the physical constraints of the joints and limbs of the human body. A typical range of motion information for the various joints is approximated by the model. In general, these conditionals can, and should, be learned from motion capture data.

Because we have chosen the local coordinate frame to be centered at the proximal joint of

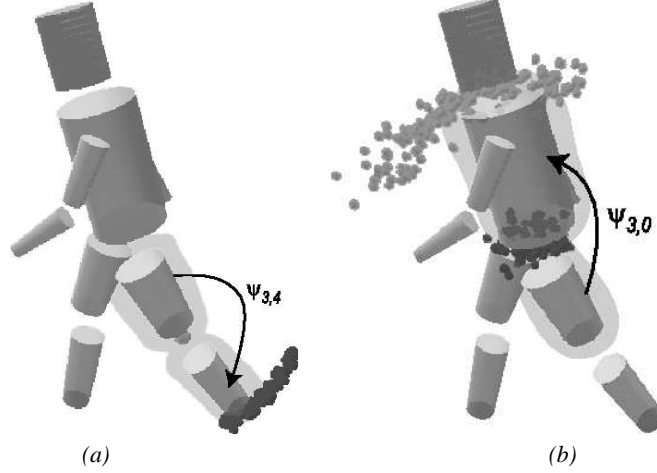

*(a)*                  *(b)*

Figure 2: *(a)* For the forwards conditional the location of part $i$ tightly constrains the proximal joint of part $j$ (light dots) while the position of the distal joint (dark dots) lies along an arc around the principal axis of rotation, approximated by a Gaussian mixture. *(b)* For the backwards conditional part $i$ constrains the *distal* joint of part $j$ (dark dots), so the proximal joint position (light dots) lies in a non-Gaussian volume again approximated using a mixture distribution.

a part, the forward and backward conditionals are not symmetric. In both directions the probability of $\mathbf{X}_j$, conditioned on $\mathbf{X}_i$, is non-Gaussian and it is approximated by a mixture of $M_{ij}$ Gaussians (typically 5-7 in the experiments here): $\psi_{ij}(\mathbf{X}_i, \mathbf{X}_j) =$

$$\lambda^0 N(\mathbf{X}_j; \mu_{ij}, \Lambda_{ij}) + (1 - \lambda^0) \sum_{m=1}^{M_{ij}} \delta_{ijm} N(\mathbf{X}_j; F_{ijm}(\mathbf{X}_i), G_{ijm}(\mathbf{X}_i)) \qquad (1)$$

where $\lambda^0$ is a fixed outlier probability, $\mu_{ij}$ and $\Lambda_{ij}$ are the mean and covariance of the Gaussian outlier process, and $F_{ijm}(.)$ and $G_{ijm}(.)$ are functions computing the mean and covariance matrix respectively of the $m$-th Gaussian mixture component. These functions allow the mean and variance of the mixture components to be function of the limb pose $\mathbf{X}_i$. $\delta_{ijm}$ is the relative weight of an individual component and $\sum_{m=1}^{M_{ij}} \delta_{ijm} = 1$.

Figure 2*a* and *b* illustrate the forward and backward conditionals respectively. For the forward case, we examine the distribution of calf configurations conditioned on the thigh. To illustrate the conditional distribution we sample from it and plot the endpoints of the sampled limb configurations. In the forward direction the conditional distribution over $\mathbf{x}_j$ (the position of the proximal joint of part $j$) is well approximated by a Gaussian so each mixture component has the same mean and covariance for $\mathbf{x}_j$. This can be seen in the tight clustering of the light dots which lie almost on top of each other. The probability of the lower leg angle is restricted to a range of legal motions conditioned on the upper leg. This distribution over rotations is modeled by giving each mixture component a different mean rotation, $\Theta_j$, spaced evenly around the principal axis of the joint. This angular uncertainty is illustrated by the dark dots.

For the backward conditional we show the distribution over torso configurations conditioned on the thigh. In this direction the conditional predicting $\mathbf{x}_j$ (e.g. torso position) is more complicated. The location of $\mathbf{x}_i$ restricts $\mathbf{x}_j$ to lie near a hemisphere, and the orientation $\Theta_i$ and principal axis of rotation further restrict $\mathbf{x}_j$ to a strip on that hemisphere which can be seen in Figure 2*b* (light dots). Thus each mixture component in (1) is spaced evenly in $\Theta_j$ and $\mathbf{x}_j$ to represent this range of uncertainty. The combined uncertainty in torso location and orientation can be seen in the distribution of the dark dots representing the distal torso joint.

**Image Likelihoods**

The inference algorithm outlined in the next section combines the body model described above with a probabilistic image likelihood model. In particular, we define $\phi_i(\mathbf{X}_i)$ to be the likelihood of observing the image measurements conditioned on the pose of limb $i$. Ideally this model would be robust to partial occlusions, the variability of image statistics across different input sequences, and variability among subjects. To that end, we combine a variety of cues including multi-scale edge and ridge filters as well as background subtraction information. Following related work [18], the likelihoods are estimated independently for each image view by projecting the 3D model of a limb into the corresponding image projection plane. These likelihoods are then combined across views, assuming independence, and are weighted by the observability of the limb in a given view (more weight is given to views in which the limb lies parallel to the image projection plane). For more information on the formulation of the image likelihoods see [20].

## 3   Non-parametric Belief Propagation

Having defined the model it remains to specify an algorithm which will perform inference and estimate a belief distribution for each of the body parts. If it were feasible to discretize the $\mathbf{X}_i$ we could apply traditional belief propagation or a specialized inference algorithm as set out in [8]. However, the 6-dimensional configuration vector compels the use of continuous-valued random variables, and so we adopt the algorithm introduced in [12, 22] for just such types of model. It is a generalization of particle filtering [7] which allows inference over arbitrary graphs rather than just a chain. This generalization is achieved by treating the particle set which is propagated in a standard particle filter as an approximation to the "message" used in the belief propagation algorithm, and replacing the conditional distribution from the previous time step by a product of incoming message sets.

A message $m_{ij}$ from node $i \rightarrow j$ is written

$$m_{ij}(\mathbf{X}_j) = \int \psi_{ij}(\mathbf{X}_i, \mathbf{X}_j)\phi_i(\mathbf{X}_i) \prod_{k \in A_i \setminus j} m_{kj}(\mathbf{X}_i)\mathrm{d}\mathbf{X}_i, \qquad (2)$$

where $A_i$ is the set of neighbors of node $i$ and $\phi_i(\mathbf{X}_i)$ is the local likelihood associated with node $i$. The message $m_{ij}(\mathbf{X}_j)$ can be approximated by importance sampling $N' = (N-1)/M_{ij}$ times from a proposal function $f(\mathbf{X}_i)$, and then doing importance correction. (See [22] for an alternative algorithm that uses more general potential functions than the conditional distributions used here.) As discussed in [12] the samples may be stratified into groups with different proposal functions $f(\cdot)$, so some samples come from the product of all incoming messages $A_i$ into the node, some from $A_i \setminus j$ (i.e. $A_i$ excluding $j$) and some from a static importance function $Q(\mathbf{X}_i)$ — we use a limb proposal distribution based on local image measurements. For reasons of space we present only a simplified algorithm to update message $m_{ij}$ in Figure 3 which does not include the stratification but the full algorithm can be found in [12]. We use the Gibbs sampler described in [22] to form message products of $D > 2$ messages.

The algorithm must sample, evaluate, and take products over Gaussian distributions defined $\in SO(3)$ and represented in terms of unit quaternions. We adopt the approximation given in [4] for dealing with rotational distributions by treating the quaternions locally linearly in $\mathbb{R}^4$ — this approximation is only valid for kernels with small rotational covariance and can in principle suffer from singularities if product distributions are widely distributed about the sphere, but we have not encountered problems in practice.

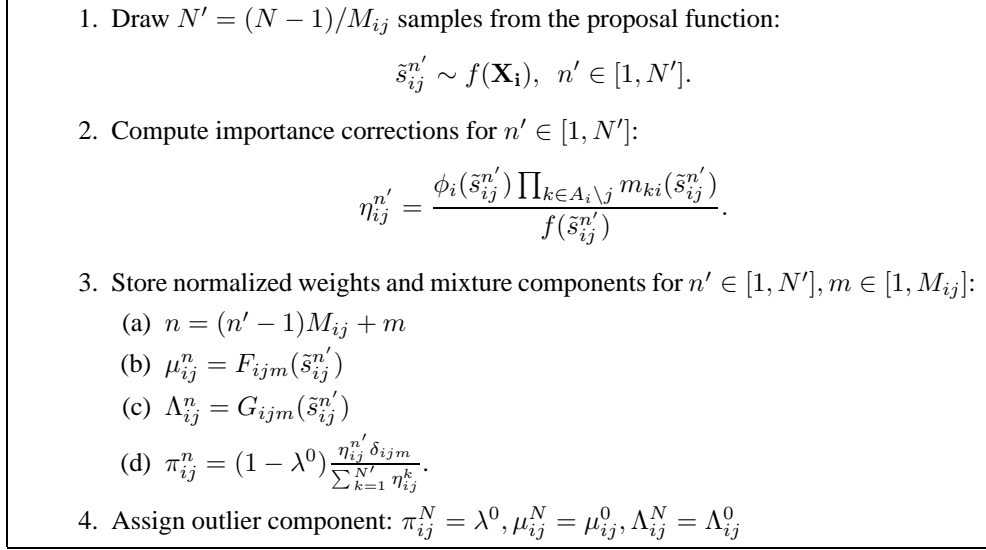

1. Draw $N' = (N-1)/M_{ij}$ samples from the proposal function:
$$\tilde{s}_{ij}^{n'} \sim f(\mathbf{X_i}), \ \ n' \in [1, N'].$$

2. Compute importance corrections for $n' \in [1, N']$:
$$\eta_{ij}^{n'} = \frac{\phi_i(\tilde{s}_{ij}^{n'}) \prod_{k \in A_i \setminus j} m_{ki}(\tilde{s}_{ij}^{n'})}{f(\tilde{s}_{ij}^{n'})}.$$

3. Store normalized weights and mixture components for $n' \in [1, N'], m \in [1, M_{ij}]$:
   (a) $n = (n'-1)M_{ij} + m$
   (b) $\mu_{ij}^n = F_{ijm}(\tilde{s}_{ij}^{n'})$
   (c) $\Lambda_{ij}^n = G_{ijm}(\tilde{s}_{ij}^{n'})$
   (d) $\pi_{ij}^n = (1 - \lambda^0)\frac{\eta_{ij}^{n'} \delta_{ijm}}{\sum_{k=1}^{N'} \eta_{ij}^k}.$

4. Assign outlier component: $\pi_{ij}^N = \lambda^0, \mu_{ij}^N = \mu_{ij}^0, \Lambda_{ij}^N = \Lambda_{ij}^0$

Figure 3: *The simplified* PAMPAS *non-parametric belief propagation algorithm.*

## 4  Experiments

We illustrate the approach by recovering 3D body pose given weak bottom-up information and clutter. The development of bottom-up part detectors is beyond the scope of this paper. Here we exploit a realistic simulation of such detectors in which: 1) the limbs are only detected 50% of the time — the remaining samples are clutter; 2) the limb detectors are non-specific in that they cannot distinguish the left and right sides of the body or the upper from lower limbs (they do, however, distinguish between legs and arms) — the result is that only a small fraction of bottom-up samples fall in the right place with the right interpretation; 3) the detectors are noisy and do not detect the limb position and orientation accurately; 4) no correct initialization samples are generated for the torso, simulating detector failure or occlusion.

Figure 4 shows results for two time instants in a video sequence taken from three calibrated cameras. After 10 iterations of belief propagation, the algorithm has discarded the samples which originated in clutter and has correctly assigned the limbs. The figure shows the initialization and the final distribution over limb poses which is computed by sampling from the belief distribution. Note that the torso is well localized even though there was no bottom-up detector for it.

## 5  Conclusion

We present a new body model and inference method that supports the goals of automatically locating and tracking an articulated body in three dimensions. We show that a "loose-limbed" model with continuous-valued parameters can effectively represent a person's location and pose, and that inference over such a model can be tractably performed using belief propagation over particle sets. Moreover, we demonstrate robust location of the person starting from imperfect initialization using a simulated body-part detector. The detector is assumed to generate both false positive initializations and false negatives; i.e. failures to detect some body parts altogether.

It is straightforward to extend the graphical model across time to implement a person

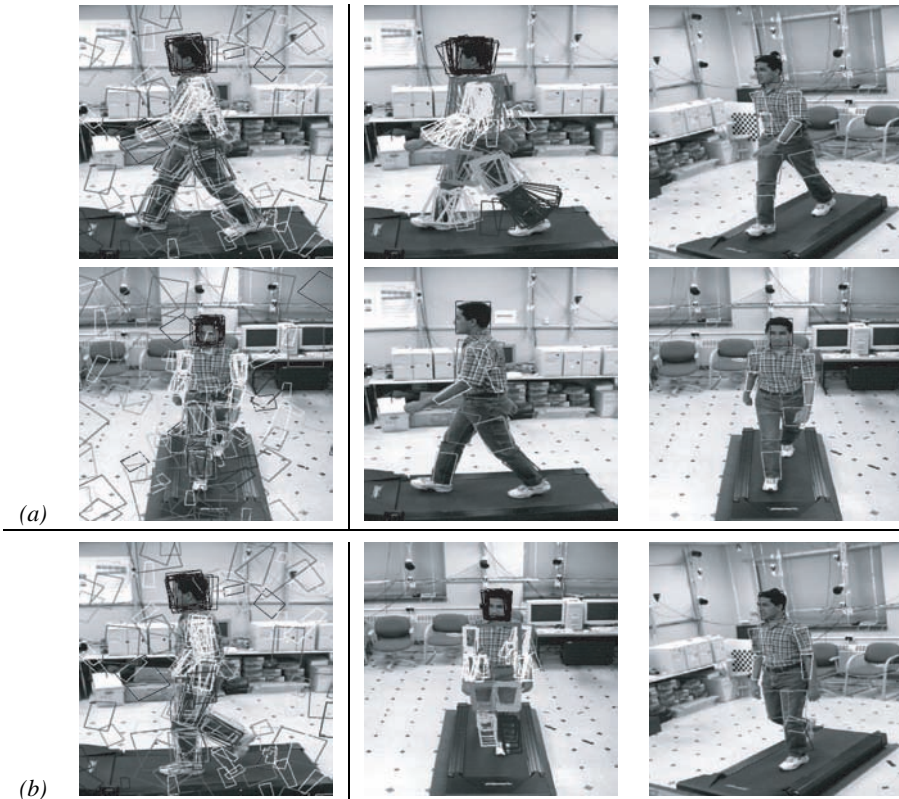

Figure 4: Inferring attractive people: Two experiments are shown; (*a*) and (*b*) show results for two different time instants in a walking cycle. Each experiment used three calibrated camera views. Left: Initialization samples drawn from noisy simulated part detectors. Part detectors are assumed to have high failure rate, generating 50% of the samples far away from any true body part. They are also non-specific; e.g. the left thigh samples are equally distributed over left and right thigh and calf. The torso is assumed to be undetectable. Right: Belief after 10 iterations of PAMPAS. We use 100 particles to model the messages between the nodes, and show 20 samples from the belief distribution, as well as the average of the top 10 percent of the belief samples as the "best" pose estimate. For brevity, (*b*) only shows the best pose from a single view.

tracker. There are several advantages of this approach compared with traditional particle filtering: the complexity of the search task is linear rather than exponential in the number of body parts; bottom-up initialization information can be incorporated in every frame; and forward-backward smoothing, either over a time-window or an entire sequence, is straight-forward.

In future work we intend to build automatic body-part detectors. Constructing reliable detectors using only low-level information (static appearance) is a challenging problem but we have the advantage of being robust to imperfect detection as noted above. We also intend to learn the conditional distributions between parts from a database of motion capture data. Together these advances should allow reliable use of the presented body model in the person tracking framework.

**Acknowledgments.** We thank Jianbo Shi for providing the image data. LS, BHS, and MJB were supported in part by the DARPA HumanID Project (ONR N000140110886).

## Footnotes

[1]Self-occlusion and self-intersection violate this assumption. These can be modeled by adding additional edges in the graph between the possibly occluding or inter-penetrating parts. In the limit this would lead to quadratic as opposed to linear computation time in the number of parts.

# References

[1] M. Burl, M. Weber and P. Perona . A probabilistic approach to object recognition using local photometry and global geometry, *ECCV*, pp. 628–641, 1998.

[2] C. Bregler and J. Malik. Tracking people with twists and exponential maps, *CVPR*, pp. 8–15, 1998.

[3] J. Coughlan and S. Ferreira. Finding deformable shapes using loopy belief propagation, *ECCV* Vol. 3, pp. 453–468, 2002.

[4] J. Deutscher, M. Isard and J. MacCormick. Automatic camera calibration from a single manhattan image, *ECCV*, pp. 175–188, 2002.

[5] J. Deutscher, A. Davison and I. Reid. Automatic partitioning of high dimensional search spaces associated with articulated body motion capture, *CVPR*, pp. 669–676, 2001.

[6] J. Deutscher, B. North, B. Bascle and A. Blake. Tracking through singularities and discontinuities by random sampling, *ICCV*, pp. 1144–1149, 1999.

[7] A. Douce, N. de Freitas and N. Gordon. Sequantial Monte Carlo methods in practice, *Statistics for Engineering and Information Sciences*, pp. 3–14, Springer Verlag, 2001.

[8] P. Felzenszwalb and D. Huttenlocher. Efficient matching of pictorial structures, *CVPR*, Vol. 2, pp. 66–73, 2000.

[9] M. Fischler and R. Elschlager. The representation and matching of pictorial structures. *IEEE. Trans. Computers*, 22(1):67–92, 1973.

[10] J. Gao and J. Shi, Inferring human upper body motion, Tech report CMU-RI-TR-03-05, 2003.

[11] S. Ioffe and D. Forsyth. Probabilistic methods for finding people, *IJCV* 43(1):45–68, 2001.

[12] M. Isard. PAMPAS: Real-valued graphical models for computer vision, *CVPR*, Vol. 1, pp. 613–620, 2003.

[13] M. Jordan, T. Sejnowski and T. Poggio. Graphical models: Foundations of neural computation, *MIT Press*, 2001.

[14] S. Ju, M. Black and Y. Yacoob. Cardboard people: A parameterized model of articulated motion. *Int. Conf. on Automatic Face and Gesture Recognition*, pp. 38–44, 1996.

[15] J. MacCormick and M. Isard. Partitioned sampling, articulated objects, and interface-quality hand tracking. *ECCV* (2), pp. 3–19, 2000.

[16] V. Pavolvić, J. Rehg, T-J. Cham and K. Murphy. A dynamic Bayesian network approach to figure tracking using learned dynamic models, *ICCV*, pp. 94–101, 1999.

[17] D. Ramanan and D. Forsyth. Finding and tracking people from the bottom up, *CVPR*, Vol. II, pp. 467–716, 2003.

[18] H. Sidenbladh and M. Black. Learning image statistics for Bayesian tracking, *ICCV*, Vol. II, pp. 709–716, 2001.

[19] H. Sidenbladh, M. Black and D. Fleet. Stochastic tracking of 3D human figures using 2D image motion, *ECCV*, vol. 2, pp. 702–718, 2000.

[20] B. Sigelman. Video-Based Tracking of 3D Human Motion Using Multiple Cameras, Brown Univ., Dept. of Comp. Sci., *Technical Report*, CS-03-08, 2003.

[21] C. Sminchisescu and B. Triggs. Covariance scaled sampling for monocular 3D body tracking, CVPR, vol. 1 pp. 447–454, 2001.

[22] E. Sudderth, A. Ihler, W. Freeman and A. Willsky. Nonparametric belief propagation, *CVPR*, Vol. 1, pp. 605–612, 2003; (see also MIT AI Lab Memo 2002-020).

[23] Y. Wu, G. Hua and T. Yu, Tracking articulated body by dynamic Markov network, ICCV, pp. 1094–1101, 2003.

[24] J. Yedidia, W. Freeman and Y. Weiss. Generalized belief propagation, *Advances in Neural Info. Proc. Sys. 13*, pp. 689–695, 2000.

[25] S. Yu, R. Gross, and J. Shi. Object segmentation by graph partitioning Concurrent object recognition and segmentation by graph partitioning, *Advances in Neural Info. Proc. Sys. 15*, pp. 1407–1414, 2003.
